# Recovering Perspective Pose with a Dual Step EM Algorithm

**Andrew D.J. Cross and Edwin R. Hancock,**
Department of Computer Science,
University of York,
York, YO1 5DD, UK.

## Abstract

This paper describes a new approach to extracting 3D perspective structure from 2D point-sets. The novel feature is to unify the tasks of estimating transformation geometry and identifying point-correspondence matches. Unification is realised by constructing a mixture model over the bi-partite graph representing the correspondence match and by effecting optimisation using the EM algorithm. According to our EM framework the probabilities of structural correspondence gate contributions to the expected likelihood function used to estimate maximum likelihood perspective pose parameters. This provides a means of rejecting structural outliers.

## 1 Introduction

The estimation of transformational geometry is key to many problems of computer vision and robotics [10]. Broadly speaking the aim is to recover a matrix representation of the transformation between image and world co-ordinate systems. In order to estimate the matrix requires a set of correspondence matches between features in the two co-ordinate systems [11]. Posed in this way there is a basic chicken-and-egg problem. Before good correspondences can be estimated, there need to be reasonable bounds on the transformational geometry. Yet this geometry is, after all, the ultimate goal of computation. This problem is usually overcome by invoking constraints to bootstrap the estimation of feasible correspondence matches [5, 8]. One of the most popular ideas is to use the epipolar constraint to prune the space of potential correspondences [5]. One of the drawbacks of this pruning strategy is that residual outliers may lead to ill-conditioned or singular parameter matrices [11].

The aim in this paper is to pose the two problems of estimating transformation geometry and locating correspondence matches using an architecture that is reminiscent of the hierarchical mixture of experts algorithm [6]. Specifically, we use a bi-partite graph to represent the current configuration of correspondence match. This graphical structure provides an architecture that can be used to gate contributions to the likelihood function for the geometric parameters using structural constraints. Correspondence matches and transformation parameters are estimated by applying the EM algorithm to the gated likelihood function. In this way we arrive at dual maximisation steps. Maximum likelihood parameters are found by minimising the structurally gated squared residuals between features in the two images being matched. Correspondence matches are updated so as to maximise the *a posteriori* probability of the observed structural configuration on the bi-partite association graph.

We provide a practical illustration in the domain of computer vision which is aimed at matching images of floppy discs under severe perspective foreshortening. However, it is important to stress that the idea of using a graphical model to provide structural constraints on parameter estimation is a task of generic importance. Although the EM algorithm has been used to extract affine and Euclidean parameters from point-sets [4] or line-sets [9], there has been no attempt to impose structural constraints of the correspondence matches. Viewed from the perspective of graphical template matching [1, 7] our EM algorithm allows an explicit deformational model to be imposed on a set of feature points. Since the method delivers statistical estimates for both the transformation parameters and their associated covariance matrix it offers significant advantages in terms of its adaptive capabilities.

## 2 Perspective Geometry

Our basic aim is to recover the perspective transformation parameters which bring a set of model or fiducial points into correspondence with their counterparts in a set of image data. Each point in the image data is represented by an augmented vector of co-ordinates $\underline{w}_i = (x_i, y_i, 1)^T$ where $i$ is the point index. The available set of image points is denoted by $\mathbf{w} = \{\underline{w}_i, \forall i \in \mathcal{D}\}$ where $\mathcal{D}$ is the point index-set. The fiducial points constituting the model are similarly represented by the set of augmented co-ordinate vectors $\mathbf{z} = \{\underline{z}_j, \forall j \in \mathcal{M}\}$. Here $\mathcal{M}$ is the index-set for the model feature-points and the $\underline{z}_j$ represent the corresponding image co-ordinates.

Perspective geometry is distinguished from the simpler Euclidean (translation, rotation and scaling) and affine (the addition of shear) cases by the presence of significant foreshortening. We represent the perspective transformation by the parameter matrix

$$\Phi^{(n)} = \begin{pmatrix} \phi_{1,1}^{(n)} & \phi_{1,2}^{(n)} & \phi_{1,3}^{(n)} \\ \phi_{2,1}^{(n)} & \phi_{2,2}^{(n)} & \phi_{2,3}^{(n)} \\ \phi_{3,1}^{(n)} & \phi_{3,2}^{(n)} & \phi_{3,3}^{(n)} \end{pmatrix} \tag{1}$$

Using homogeneous co-ordinates, the transformation between model and data is $\underline{z}_j^{(n)} = (\frac{1}{\underline{z}_j^T \cdot \Psi^{(n)}})^{-1} \Phi^{(n)} \underline{z}_j$, where $\Psi^{(n)} = (\phi_{3,1}^{(n)}, \phi_{3,2}^{(n)}, 1)^T$ is a column-vector formed from the elements in bottom row of the transformation matrix.

## 3   Relational Constraints

One of our goals in this paper is to exploit structural constraints to improve the recovery of perspective parameters from sets of feature points. We abstract the process as bi-partite graph matching. Because of its well documented robustness to noise and change of viewpoint, we adopt the Delaunay triangulation as our basic representation of image structure [3]. We establish Delaunay triangulations on the data and the model, by seeding Voronoi tessellations from the feature-points.

The process of Delaunay triangulation generates relational graphs from the two sets of point-features. More formally, the point-sets are the nodes of a data graph $G_D = \{\mathcal{D}, E_D\}$ and a model graph $G_M = \{\mathcal{M}, E_M\}$. Here $E_D \subseteq \mathcal{D} \times \mathcal{D}$ and $E_M \subseteq \mathcal{M} \times \mathcal{M}$ are the edge-sets of the data and model graphs. Key to our matching process is the idea of using the edge-structure of Delaunay graphs to constrain the correspondence matches between the two point-sets. This correspondence matching is denoted by the function $f : \mathcal{M} \to \mathcal{D}$ from the nodes of the data-graph to those of the model graph. According to this notation the statement $f^{(n)}(i) = j$ indicates that there is a match between the node $i \in \mathcal{D}$ of the model-graph to the node $j \in \mathcal{M}$ of the model graph at iteration $n$ of the algorithm. We use the binary indicator

$$s_{i,j}^{(n)} = \begin{cases} 1 & \text{if } f^{(n)}(i) = j \\ 0 & \text{otherwise} \end{cases} \tag{2}$$

to represent the configuration of correspondence matches.

We exploit the structure of the Delaunay graphs to compute the consistency of match using the Bayesian framework for relational graph-matching recently reported by Wilson and Hancock [12]. Suffice to say that consistency of a configuration of matches residing on the neighbourhood $R_i = i \cup \{k \ ; \ (i,k) \in E_D\}$ of the node $i$ in the data-graph and its counterpart $S_j = j \cup \{l \ ; \ (j,l) \in E_m\}$ for the node $j$ in the model-graph is gauged by Hamming distance. The Hamming distance $H(i,j)$ counts the number of matches on the data-graph neighbourhood $R_i$ that are inconsistently matched onto the model-graph neighbourhood $S_j$. According to Wilson and Hancock [12] the structural probability for the correspondence match $f(i) = j$ at iteration $n$ of the algorithm is given by

$$\zeta_{i,j}^{(n)} = \frac{\exp\left[-\beta H(i,j)\right]}{\sum_{j \in \mathcal{M}} \exp\left[-\beta H(i,j)\right]} \tag{3}$$

In the above expression, the Hamming distance is given by $H(i,j) = \sum_{(k,l) \in R_i \bullet S_j} (1 - s_{k,l}^{(n)})$ where the symbol $\bullet$ denotes the composition of the data-graph relation $R_i$ and the model-graph relation $S_j$. The exponential constant $\beta = \ln \frac{1 - P_e}{P_e}$ is related to the uniform probability of structural matching errors $P_e$. This probability is set to reflect the overlap of the two point-sets. In the work reported here we set $P_e = \frac{2\|\mathcal{M}\| - \|\mathcal{D}\|}{\|\mathcal{M}\| + \|\mathcal{D}\|}$.

## 4   The EM Algorithm

Our aim is to extract perspective pose parameters and correspondences matches from the two point-sets using the EM algorithm. According to the original work

of Dempster, Laird and Rubin [2] the expected likelihood function is computed by weighting the current log-probability density by the *a posteriori* measurement probabilities computed from the preceding maximum likelihood parameters. Jordan and Jacobs [6] augment the process with a graphical model which effectively gates contributions to the expected log-likelihood function. Here we provide a variant of this idea in which the bi-partite graph representing the correspondences matches gate the log-likelihood function for the perspective pose parameters.

## 4.1  Mixture Model

Our basic aim is to jointly maximize the data-likelihood $p(\mathbf{w}|\mathbf{z}, f, \Phi)$ over the space of correspondence matches $f$ and the matrix of perspective parameters $\Phi$. To commence our development, we assume observational independence and factorise the conditional measurement density over the set of data-items

$$p(\mathbf{w}|\mathbf{z}, f, \Phi) = \prod_{i \in \mathcal{D}} p(\underline{w}_i | \mathbf{z}, f, \Phi) \tag{4}$$

In order to apply the apparatus of the EM algorithm to maximising $p(\mathbf{w}|\mathbf{z}, f, \Phi)$ with respect to $f$ and $\Phi$, we must establish a mixture model over the space of correspondence matches. Accordingly, we apply Bayes theorem to expand over the space of match indicator variables. In other words,

$$p(\underline{w}_i | \mathbf{z}, f, \Phi) = \sum_{s_{i,j} \in f} p(\underline{w}_i, s_{i,j} | \mathbf{z}, f, \Phi) \tag{5}$$

In order to develop a tractable likelihood function, we apply the chain rule of conditional probability. In addition, we use the indicator variables to control the switching of the conditional measurement densities via exponentiation. In other words we assume $p(\underline{w}_i | s_{i,j}, \underline{z}_j, \Phi) = p(\underline{w}_i | \underline{z}_j, \Phi)^{s_{i,j}}$.

With this simplification,the mixture model for the correspondence matching process leads to the following expression for the expected likelihood function

$$Q(f^{(n+1)}, \Phi^{(n+1)} | f^{(n)}, \Phi^{(n)}) = \sum_{i \in \mathcal{D}} \sum_{i \in \mathcal{M}} P(s_{i,j} | \mathbf{w}, \mathbf{z}, f^{(n)}, \Phi^{(n)}) s_{i,j}^{(n)} \ln p(\underline{w}_i | \underline{z}_j, \Phi^{(n+1)})$$
$$\tag{6}$$

To further simplify matters we make a mean-field approximation and replace $s_{i,j}^{(n)}$ by its average value, i.e. we make use of the fact that $E(s_{i,j}^{(n)}) = \zeta_{i,j}^{(n)}$. In this way the structural matching probabilities gate contributions to the expected likelihood function. This mean-field approximation alleviates problems associated with local optima which are likely to occur if the likelihood function is discretised by gating with $s_{i,j}$.

## 4.2  Expectation

Using the Bayes rule, we can re-write the *a posteriori* measurement probabilities in terms of the components of the conditional measurement densities appearing in the mixture model in equation (5)

$$P(s_{i,j} | \mathbf{w}, \mathbf{z}, f^{(n)}, \Phi^{(n+1)}) = \frac{\zeta_{i,j}^{(n)} p(\underline{w}_i | \underline{z}_j, \Phi^{(n)})}{\sum_{j' \in \mathcal{M}} \zeta_{i,j'}^{(n)} p(\underline{w}_i | \underline{z}_{j'}, \Phi^{(n)})} \tag{7}$$

In order to proceed with the development of a point registration process we require a model for the conditional measurement densities, i.e. $p(\underline{w}_i | \underline{z}_j, \Phi^{(n)})$. Here we assume that the required model can be specified in terms of a multivariate Gaussian distribution. The random variables appearing in these distributions are the error residuals for the position predictions of the $j$th model line delivered by the current estimated transformation parameters. Accordingly we write

$$p(\underline{w}_i | \underline{z}_j, \Phi^{(n)}) = \frac{1}{(2\pi)^{\frac{3}{2}} \sqrt{|\Sigma|}} \exp\left[-\frac{1}{2}(\underline{w}_i - \underline{z}_j^{(n)})^T \Sigma^{-1} (\underline{w}_i - \underline{z}_j^{(n)})\right] \qquad (8)$$

In the above expression $\Sigma$ is the variance-covariance matrix for the vector of error-residuals $\epsilon_{i,j}(\Phi^{(n)}) = \underline{w}_i - \underline{z}_j^{(n)}$ between the components of the predicted measurement vectors $\underline{z}_j'$ and their counterparts in the data, i.e. $\underline{w}_i$. Formally, the matrix is related to the expectation of the outer-product of the error-residuals i.e. $\Sigma = E[\epsilon_{i,j}(\Phi^{(n)})\epsilon_{i,j}(\Phi^{(n)})^T]$.

## 4.3  Maximisation

The maximisation step of our matching algorithm is based on two coupled update processes. The first of these aims to locate maximum *a posteriori* probability correspondence matches. The second class of update operation is concerned with locating maximum likelihood transformation parameters. We effect the coupling by allowing information flow between the two processes. Correspondences located by maximum *a posteriori* graph-matching are used to constrain the recovery of maximum likelihood transformation parameters. *A posteriori* measurement probabilities computed from the updated transformation parameters are used to refine the correspondence matches.

In terms of the indicator variables matches the configuration of maximum *a posteriori* probability correspondence matches is updated as follows

$$f^{(n+1)}(i) = \arg\max_{j\in\mathcal{M}} P(\underline{z}_j | \underline{w}_i, \Phi^{(n)}) \frac{\exp\left[-\beta \sum_{(k,l)\in R_i \bullet S_j} (1 - s_{k,l}^{(n)})\right]}{\sum_{j\in\mathcal{M}} \exp\left[-\beta \sum_{(k,l)\in R_i \bullet S_j} (1 - s_{k,l}^{(n)})\right]} \qquad (9)$$

The maximum likelihood transformation parameters satisfy the condition

$$\Phi^{(n+1)} = \arg\min_{\Phi} \sum_{i\in\mathcal{D}} \sum_{i\in\mathcal{M}} P(\underline{z}_j | \underline{w}_i, \Phi^{(n)}) \zeta_{i,j}^{(n)} (\underline{w}_i - \underline{z}_j^{(n)})^T \Sigma^{-1} (\underline{w}_i - \underline{z}_j^{(n)}) \qquad (10)$$

In the case of perspective geometry where we have used homogeneous co-ordinates the saddle-point equations are not readily amenable in a closed-form linear fashion. Instead, we solve the non-linear maximisation problem using the Levenberg-Marquardt technique. This non-linear optimisation technique offers a compromise between the steepest gradient and inverse Hessian methods. The former is used when close to the optimum while the latter is used far from it.

## 5  Experiments

The real-world evaluation of our matching method is concerned with recognising planer objects in different 3D poses. The object used in this study is a 3.5 inch

floppy disk which is placed on a desktop. The scene is viewed with a low-quality SGI IndyCam. The feature points used to triangulate the object are corners. Since the imaging process is not accurately modelled by a perspective transformation under pin-hole optics, the example provides a challenging test of our matching process.

Our experiments are illustrated in Figure 1. The first two columns show the views under match. In the first example (the upper row of Figure 1) we are concerned with matching when there is a significant difference in perspective forshortening. In the example shown in the lower row of Figure 1, there is a rotation of the object in addition to the foreshortening. The images in the third column are the initial matching configurations. Here the perspective parameter matrix has been selected at random. The fourth column in Figure 1 shows the final matching configuration after the EM algorithm has converged. In both cases the final registration is accurate. The algorithm appears to be capable of recovering good matches even when the initial pose estimate is poor.

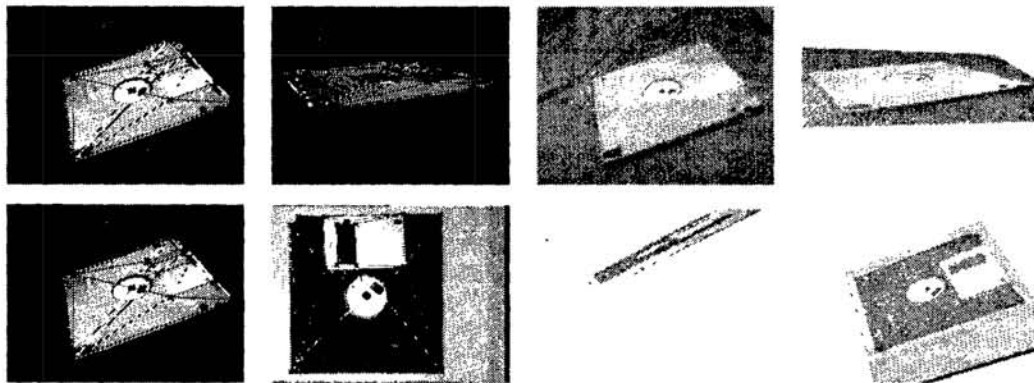

Figure 1: Images Under Match, Initial and Final Configurations.

We now turn to measuring the sensitivity of our method. In order to illustrate the benefits offered by the structural gating process, we compare its performance with a conventional least-squares parameter estimation process. Figure 2 shows a comparison of the two algorithms for a problem involving a point-set of 20 nodes. Here we show the RMS error as a function of the number of points which have correct correspondence matches. The break-even point occurs when 8 nodes are initially matched correctly and there are 12 errors. Once the number of initially correct correspondences exceeds 8 then the EM method consistently outperforms the least-squares estimation.

## 6   Conclusions

Our main contributions in this paper are twofold. The theoretical contribution has been to develop a mixture model that allows a graphical structure to to constrain the estimation of maximum likelihood model parameters. The second contribution is a practical one, and involves the application of the mixture model to the estimation of perspective pose parameters. There are a number of ways in which the ideas developed in this paper can be extended. For instance, the framework is readily extensible to the recognition of more complex non-planar objects.

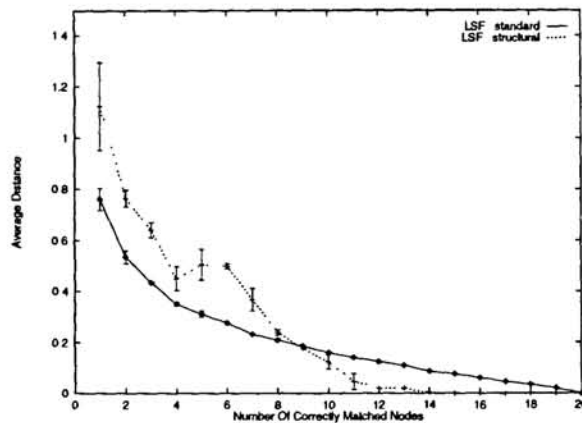

Figure 2: Structural Sensitivity.

## References

[1] Y. Amit and A. Kong, "Graphical Templates for Model Registration", *IEEE PAMI*, **18**, pp. 225–236, 1996.

[2] A.P. Dempster, Laird N.M. and Rubin D.B., "Maximum-likelihood from incomplete data via the EM algorithm", J. Royal Statistical Soc. Ser. B (methodological),**39**, pp 1-38, 1977.

[3] O.D. Faugeras, E. Le Bras-Mehlman and J-D. Boissonnat, "Representing Stereo Data with the Delaunay Triangulation", *Artificial Intelligence*, **44**, pp. 41–87, 1990.

[4] S. Gold, Rangarajan A. and Mjolsness E., "Learning with pre-knowledge: Clustering with point and graph-matching distance measures", *Neural Computation*, **8**, pp. 787–804, 1996.

[5] R.I. Hartley, "Projective Reconstruction and Invariants from Multiple Images", *IEEE PAMI*, **16**, pp. 1036—1041, 1994.

[6] M.I. Jordan and R.A. Jacobs, "Hierarchical Mixtures of Experts and the EM Algorithm", *Neural Computation*, **6**, pp. 181-214, 1994.

[7] M. Lades, J.C. Vorbruggen, J. Buhmann, J. Lange, C. von der Maalsburg, R.P. Wurtz and W.Konen, "Distortion-invariant object-recognition in a dynamic link architecture", *IEEE Transactions on Computers*, **42**, pp. 300–311, 1993

[8] D.P. McReynolds and D.G. Lowe, "Rigidity Checking of 3D Point Correspondences under Perspective Projection", *IEEE PAMI*, **18** , pp. 1174–1185, 1996.

[9] S. Moss and E.R. Hancock, "Registering Incomplete Radar Images with the EM Algorithm", *Image and Vision Computing*, **15**, 637–648, 1997.

[10] D. Oberkampf, D.F. DeMenthon and L.S. Davis, "Iterative Pose Estimation using Coplanar Feature Points", *Computer Vision and Image Understanding*, **63**, pp. 495–511, 1996.

[11] P. Torr, A. Zisserman and S.J. Maybank, "Robust Detection of Degenerate Configurations for the Fundamental Matrix", *Proceedings of the Fifth International Conference on Computer Vision*, pp. 1037–1042, 1995.

[12] R.C. Wilson and E.R. Hancock, "Structural Matching by Discrete Relaxation", *IEEE PAMI*, **19**, pp.634-648, 1997.
